# Prediction of Protein Topologies Using Generalized IOHMMs and RNNs

**Gianluca Pollastri and Pierre Baldi**
Department of Information and Computer Science
University of California, Irvine
Irvine, CA 92697-3425
*gpollast,pfbaldi@ics.uci.edu*

**Alessandro Vullo and Paolo Frasconi**
Dipartimento di Sistemi e Informatica
Università di Firenze
Via di Santa Marta 3, 50139 Firenze, ITALY
*vullo,paolo@dsi.unifi.it*

## Abstract

We develop and test new machine learning methods for the prediction of topological representations of protein structures in the form of coarse- or fine-grained contact or distance maps that are translation and rotation invariant. The methods are based on generalized input-output hidden Markov models (GIOHMMs) and generalized recursive neural networks (GRNNs). The methods are used to predict topology directly in the fine-grained case and, in the coarse-grained case, indirectly by first learning how to score candidate graphs and then using the scoring function to search the space of possible configurations. Computer simulations show that the predictors achieve state-of-the-art performance.

## 1  Introduction: Protein Topology Prediction

Predicting the 3D structure of protein chains from the linear sequence of amino acids is a fundamental open problem in computational molecular biology [1]. Any approach to the problem must deal with the basic fact that protein structures are translation and rotation invariant. To address this invariance, we have proposed a machine learning approach to protein structure prediction [4] based on the prediction of topological representations of proteins, in the form of contact or distance maps. The contact or distance map is a 2D representation of neighborhood relationships consisting of an adjacency matrix at some distance cutoff (typically in the range of 6 to 12 Å), or a matrix of pairwise Euclidean distances. Fine-grained maps are derived at the amino acid or even atomic level. Coarse maps are obtained by looking at secondary structure elements, such as helices, and the distance between their centers of gravity or, as in the simulations below, the minimal distances between their $C_\alpha$ atoms. Reasonable methods for reconstructing 3D coordinates from contact/distance maps have been developed in the NMR literature and elsewhere

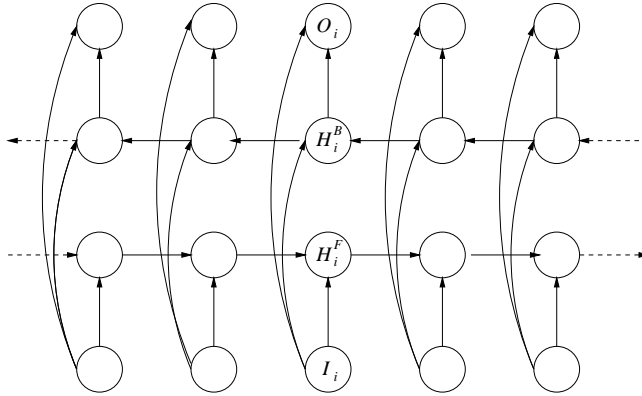

Figure 1: Bayesian network for bidirectional IOHMMs consisting of input units, output units, and both forward and backward Markov chains of hidden states.

[14] using distance geometry and stochastic optimization techniques. Thus the main focus here is on the more difficult task of contact map prediction.

Various algorithms for the prediction of contact maps have been developed, in particular using feedforward neural networks [6]. The best contact map predictor in the literature and at the last CASP prediction experiment reports an average precision [True Positives/(True Positives + False Positives)] of 21% for distant contacts, i.e. with a linear distance of 8 amino acid or more [6] for fine-grained amino acid maps. While this result is encouraging and well above chance level by a factor greater than 6, it is still far from providing sufficient accuracy for reliable 3D structure prediction. A key issue in this area is the amount of noise that can be tolerated in a contact map prediction without compromising the 3D-reconstruction step. While systematic tests in this area have not yet been published, preliminary results appear to indicate that recovery of as little as half of the distant contacts may suffice for proper reconstruction, at least for proteins up to 150 amino acid long (Rita Casadio and Piero Fariselli, private communication and oral presentation during CASP4 [10]).

It is important to realize that the input to a fine-grained contact map predictor need not be confined to the sequence of amino acids only, but may also include evolutionary information in the form of profiles derived by multiple alignment of homologue proteins, or structural feature information, such as secondary structure (alpha helices, beta strands, and coils), or solvent accessibility (surface/buried), derived by specialized predictors [12, 13]. In our approach, we use different GIOHMM and GRNN strategies to predict both structural features and contact maps.

## 2 GIOHMM Architectures

Loosely speaking, GIOHMMs are Bayesian networks with input, hidden, and output units that can be used to process complex data structures such as sequences, images, trees, chemical compounds and so forth, built on work in, for instance, [5, 3, 7, 2, 11]. In general, the connectivity of the graphs associated with the hidden units matches the structure of the data being processed. Often multiple copies of the same hidden graph, but with different edge orientations, are used in the hidden layers to allow *direct* propagation of information in all relevant directions.

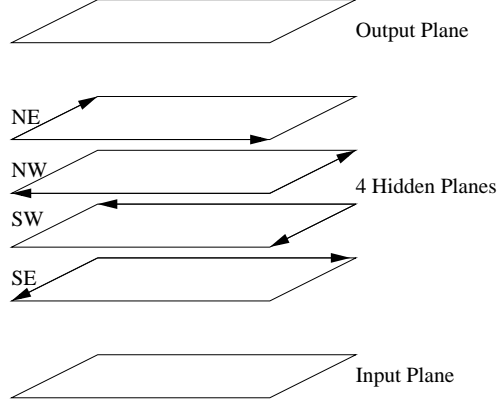

Figure 2: 2D GIOHMM Bayesian network for processing two-dimensional objects such as contact maps, with nodes regularly arranged in one input plane, one output plane, and four hidden planes. In each hidden plane, nodes are arranged on a square lattice, and all edges are oriented towards the corresponding cardinal corner. Additional directed edges run vertically in column from the input plane to each hidden plane, and from each hidden plane to the output plane.

To illustrate the general idea, a first example of GIOHMM is provided by the bidirectional IOHMMs (Figure 1) introduced in [2] to process sequences and predict protein structural features, such as secondary structure. Unlike standard HMMs or IOHMMS used, for instance in speech recognition, this architecture is based on two hidden markov chains running in opposite directions to leverage the fact that biological sequences are spatial objects rather than temporal sequences. Bidirectional IOHMMs have been used to derive a suite of structural feature predictors [12, 13, 4] available through http://promoter.ics.uci.edu/BRNN-PRED/. These predictors have accuracy rates in the 75-80% range on a per amino acid basis.

## 2.1 Direct Prediction of Topology

To predict contact maps, we use a 2D generalization of the previous 1D Bayesian network. The basic version of this architecture (Figures 2) contains 6 layers of units: input, output, and four hidden layers, one for each cardinal corner. Within each column indexed by $i$ and $j$, connections run from the input to the four hidden units, and from the four hidden units to the output unit. In addition, the hidden units in each hidden layer are arranged on a square or triangular lattice, with all the edges oriented towards the corresponding cardinal corner. Thus the parameters of this two-dimensional GIOHMMs, in the square lattice case, are the conditional probability distributions:

$$
\begin{cases}
P(O_i|I_{i,j}, H_{i,j}^{NE}, H_{i,j}^{NW}, H_{i,j}^{SW}, H_{i,j}^{SE}) \\
P(H_{i,j}^{NE}|I_{i,j}, H_{i-1,j}^{NE}, H_{i,j-1}^{NE}) \\
P(H_{i,j}^{NW}|I_{i,j}, H_{i+1,j}^{NW}, H_{i,j-1}^{NW}) \\
P(H_{i,j}^{SW}|I_{i,j}, H_{i+1,j}^{SW}, H_{i,j+1}^{SW}) \\
P(H_{i,j}^{SE}|I_{i,j}, H_{i-1,j}^{SE}, H_{i,j+1}^{SE})
\end{cases}
\tag{1}
$$

In a contact map prediction at the amino acid level, for instance, the $(i,j)$ output represents the probability of whether amino acids $i$ and $j$ are in contact or not.

This prediction depends directly on the $(i, j)$ input and the four-hidden units in the same column, associated with omni-directional contextual propagation in the hidden planes. In the simulations reported below, we use a more elaborated input consisting of a $20 \times 20$ probability matrix over amino acid pairs derived from a multiple alignment of the given protein sequence and its homologues, as well as the structural features of the corresponding amino acids, including their secondary structure classification and their relative exposure to the solvent, derived from our corresponding predictors.

It should be clear how GIOHMM ideas can be generalized to other data structures and problems in many ways. In the case of 3D data, for instance, a standard GIOHMM would have an input cube, an output cube, and up to 8 cubes of hidden units, one for each corner with connections inside each hidden cube oriented towards the corresponding corner. In the case of data with an underlying tree structure, the hidden layers would correspond to copies of the same tree with different orientations and so forth. Thus a fundamental advantage of GIOHMMs is that they can process a wide range of data structures of variable sizes and dimensions.

## 2.2   Indirect Prediction of Topology

Although GIOHMMs allow flexible integration of contextual information over ranges that often exceed what can be achieved, for instance, with fixed-input neural networks, the models described above still suffer from the fact that the connections remain local and therefore long-ranged propagation of information during learning remains difficult. Introduction of large numbers of long-ranged connections is computationally intractable but in principle not necessary since the number of contacts in proteins is known to grow *linearly* with the length of the protein, and hence connectivity is inherently sparse. The difficulty of course is that the location of the long-ranged contacts is not known.

To address this problem, we have developed also a complementary GIOHMM approach described in Figure 3 where a candidate graph structure is proposed in the hidden layers of the GIOHMM, with the two different orientations naturally associated with a protein sequence. Thus the hidden graphs change with each protein. In principle the output ought to be a single unit (Figure 3b) which directly computes a global score for the candidate structure presented in the hidden layer. In order to cope with long-ranged dependencies, however, it is preferable to compute a set of local scores (Figure 3c), one for each vertex, and combine the local scores into a global score by averaging.

More specifically, consider a true topology represented by the undirected contact graph $G^* = (V, E^*)$, and a candidate undirected prediction graph $G = (V, E)$. A global measure of how well $E$ approximates $E^*$ is provided by the information-retrieval $F_1$ score defined by the normalized edge-overlap $F_1 = 2|E \cap E^*|/(|E| + |E^*|) = 2PR/(P + R)$, where $P = |E \cap E^*|/|E|$ is the precision (or specificity) and $R = |E \cap E^*|/|E^*|$ is the recall (or sensitivity) measure. Obviously, $0 \leq F_1 \leq 1$ and $F_1 = 1$ if and only if $E = E^*$. The scoring function $F_1$ has the property of being monotone in the sense that if $|E| = |E'|$ then $F_1(E) < F_1(E')$ if and only if $|E \cap E^*| < |E' \cap E^*|$. Furthermore, if $E' = E \cup \{e\}$ where $e$ is an edge in $E^*$ but not in $E$, then $F_1(E') > F_1(E)$. Monotonicity is important to guide the search in the space of possible topologies. It is easy to check that a simple search algorithm based on $F_1$ takes on the order of $O(|V|^3)$ steps to find $E^*$, basically by trying all possible edges one after the other. The problem then is to learn $F_1$, or rather a good approximation to $F_1$.

To approximate $F_1$, we first consider a similar local measure $F_v$ by considering the

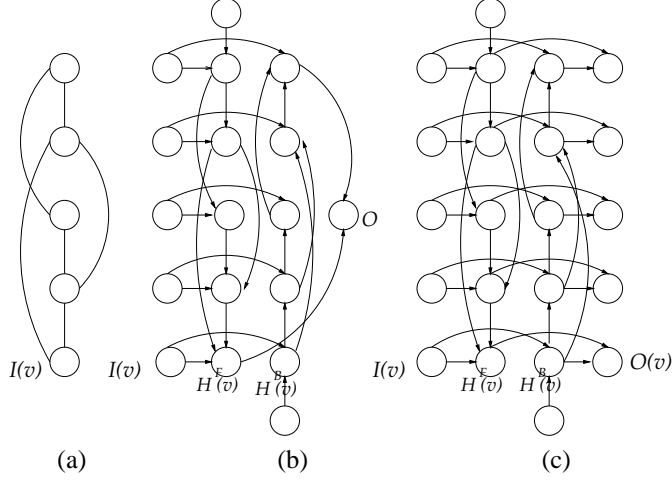

Figure 3: Indirect prediction of contact maps. (a) target contact graph to be predicted. (b) GIOHMM with two hidden layers: the two hidden layers correspond to two copies of the same candidate graph oriented in opposite directions from one end of the protein to the other end. The single output $O$ is the global score of how well the candidate graph approximates the true contact map. (c) Similar to (b) but with a local score $O(v)$ at each vertex. The local scores can be averaged to produce a global score. In (b) and (c) $I(v)$ represents the input for vertex $v$, and $H^F(v)$ and $H^B(v)$ are the corresponding hidden variables.

set $E_v$ of edges adjacent to vertex $v$ and $F_v = 2|E_v \cap E_v^*|/(|E_v| + |E_v^*|)$ with the global average $\bar{F} = \sum_v F_v/|V|$. If $n$ and $n^*$ are the average degrees of $G$ and $G^*$, it can be shown that:

$$F_1 = \frac{1}{|V|} \sum_v \frac{2|E_v \cap E^*|}{n + n^*} \quad \text{and} \quad \bar{F} = \frac{1}{|V|} \sum_v \frac{2|E_v \cap E^*|}{n + \epsilon_v + n^* + \epsilon_v^*} \tag{2}$$

where $n + \epsilon_v$ (resp. $n^* + \epsilon_v^*$) is the degree of $v$ in $G$ (resp. in $G^*$). In particular, if $G$ and $G^*$ are regular graphs, then $F_1(E) = \bar{F}(E)$ so that $\bar{F}$ is a good approximation to $F_1$. In the contact map regime where the number of contacts grows linearly with the length of the sequence, we should have in general $|E| \approx |E^*| \approx (1 + \alpha)|V|$ so that each node on average has $n = n^* = 2(1 + \alpha)$ edges. The value of $\alpha$ depends of course on the neighborhood cutoff.

As in reinforcement learning, to learn the scoring function one is faced with the problem of generating good training sets in a high dimensional space, where the states are the topologies (graphs), and the policies are algorithms for adding a single edge to a given graph. In the simulations we adopt several different strategies including static and dynamic generation. Within dynamic generation we use three exploration strategies: random exploration (successor graph chosen at random), pure exploitation (successor graph maximizes the current scoring function), and semi-uniform exploitation to find a balance between exploration and exploitation [with probability $\epsilon$ (resp. $1 - \epsilon$) we choose random exploration (resp. pure exploitation)].

## 3 GRNN Architectures

Inference and learning in the protein GIOHMMs we have described is computationally intensive due to the large number of undirected loops they contain. This problem can be addressed using a neural network reparameterization assuming that: (a) all the nodes in the graphs are associated with a deterministic vector (note that in the case of the output nodes this vector can represent a probability distribution so that the overall model remains probabilistic); (b) each vector is a deterministic function of its parents; (c) each function is parameterized using a neural network (or some other class of approximators); and (d) weight-sharing or stationarity is used between similar neural networks in the model. For example, in the 2D GIOHMM contact map predictor, we can use a total of 5 neural networks to recursively compute the four hidden states and the output in each column in the form:

$$
\begin{cases}
O_{ij} = \mathcal{N}_O(I_{ij}, H_{i,j}^{NW}, H_{i,j}^{NE}, H_{i,j}^{SW}, H_{i,j}^{SE}) \\
H_{i,j}^{NE} = \mathcal{N}_{NE}(I_{i,j}, H_{i-1,j}^{NE}, H_{i,j-1}^{NE}) \\
H_{i,j}^{NW} = \mathcal{N}_{NW}(I_{i,j}, H_{i+1,j}^{NW}, H_{i,j-1}^{NW}) \\
H_{i,j}^{SW} = \mathcal{N}_{SW}(I_{i,j}, H_{i+1,j}^{SW}, H_{i,j+1}^{SW}) \\
H_{i,j}^{SE} = \mathcal{N}_{SE}(I_{i,j}, H_{i-1,j}^{SE}, H_{i,j+1}^{SE})
\end{cases}
\tag{3}
$$

In the NE plane, for instance, the boundary conditions are set to $H_{ij}^{NE} = 0$ for $i = 0$ or $j = 0$. The activity vector associated with the hidden unit $H_{ij}^{NE}$ depends on the local input $I_{ij}$, and the activity vectors of the units $H_{i-1,j}^{NE}$ and $H_{i,j-1}^{NE}$. Activity in NE plane can be propagated row by row, West to East, and from the first row to the last (from South to North), or column by column South to North, and from the first column to the last. These GRNN architectures can be trained by gradient descent by unfolding the structures in space, leveraging the acyclic nature of the underlying GIOHMMs.

## 4 Data

Many data sets are available or can be constructed for training and testing purposes, as described in the references. The data sets used in the present simulations are extracted from the publicly available Protein Data Bank (PDB) and then redundancy reduced, or from the non-homologous subset of PDB Select (ftp://ftp.embl-heidelberg.de/pub/databases/). In addition, we typically exclude structures with poor resolution (less than 2.5-3 Å), sequences containing less than 30 amino acids, and structures containing multiple sequences or sequences with chain breaks. For coarse contact maps, we use the DSSP program [9] (CMBI version) to assign secondary structures and we remove also sequences for which DSSP crashes. The results we report for fine-grained contact maps are derived using 424 proteins with lengths in the 30-200 range for training and an additional non-homologous set of 48 proteins in the same length range for testing. For the coarse contact map, we use a set of 587 proteins of length less than 300. Because the average length of a secondary structure element is slightly above 7, the size of a coarse map is roughly 2% the size of the corresponding amino acid map.

## 5 Simulation Results and Conclusions

We have trained several 2D GIOHMM/GRNN models on the direct prediction of fine-grained contact maps. Training of a single model typically takes on the order of a week on a fast workstation. A sample of validation results is reported in Table 1 for four different distance cutoffs. Overall percentages of correctly predicted contacts

Table 1: Direct prediction of amino acid contact maps. Column 1: four distance cutoffs. Column 2, 3, and 4: overall percentages of amino acids correctly classified as contacts, non-contacts, and in total. Column 5: Precision percentage for distant contacts ($|i - j| \geq 8$) with a threshold of 0.5. Single model results except for last line corresponding to an ensemble of 5 models.

| Cutoff | Contact | Non-Contact | Total | Precision (P) |
|--------|---------|-------------|-------|---------------|
| 6 Å    | .714    | .998        | .985  | .594          |
| 8 Å    | .638    | .998        | .970  | .670          |
| 10 Å   | .512    | .993        | .931  | .557          |
| 12 Å   | .433    | .987        | .878  | .549          |
| 12 Å   | .445    | .990        | .883  | .717          |

and non-contacts at all linear distances, as well as precision results for distant contacts ($|i - j| \geq 8$) are reported for a single GIOHMM/GRNN model. The model has $k = 14$ hidden units in the hidden and output layers of the four hidden networks, as well as in the hidden layer of the output network. In the last row, we also report as an example the results obtained at $12\mathring{A}$ by an ensemble of 5 networks with $k = 11, 12, 13, 14$ and 15. Note that precision for distant contacts exceeds all previously reported results and is well above 50%.

For the prediction of coarse-grained contact maps, we use the indirect GIOHMM/GRNN strategy and compare different exploration/exploitation strategies: random exploration, pure exploitation, and their convex combination (semi-uniform exploitation). In the semi-uniform case we set the probability of random uniform exploration to $\epsilon = 0.4$. In addition, we also try a fourth hybrid strategy in which the search proceeds greedily (i.e. the best successor is chosen at each step, as in pure exploitation), but the network is trained by randomly sub-sampling the successors of the current state. Eight numerical features encode the input label of each node: one-hot encoding of secondary structure classes; normalized linear distances from the N to C terminus; average, maximum and minimum hydrophobic character of the segment (based on the Kyte-Doolittle scale with a moving window of length 7). A sample of results obtained with 5-fold cross-validation is shown in Table 2. Hidden state vectors have dimension $k = 5$ with no hidden layers. For each strategy we measure performances by means of several indices: micro and macro-averaged precision ($mP$, $MP$), recall ($mR$, $MR$) and $F_1$ measure ($mF_1$, $MF_1$). Micro-averages are derived based on each pair of secondary structure elements in each protein, whereas macro-averages are obtained on a per-protein basis, by first computing precision and recall for each protein, and then averaging over the set of all proteins. In addition, we also measure the micro and macro averages for specificity in the sense of percentage of correct prediction for non-contacts ($mP(nc)$, $MP(nc)$). Note the tradeoffs between precision and recall across the training methods, the hybrid method achieving the best $F1$ results.

Table 2: Indirect prediction of coarse contact maps with dynamic sampling.

| Strategy | $mP$ | $mP(nc)$ | $mR$ | $mF_1$ | $MP$ | $MP(nc)$ | $MR$ | $MF_1$ |
|----------|------|----------|------|--------|------|----------|------|--------|
| Random exploration | .715 | .769 | .418 | .518 | .767 | .709 | .469 | .574 |
| Semi-uniform | .454 | .787 | .631 | .526 | .507 | .767 | .702 | .588 |
| Pure exploitation | .431 | .806 | .726 | .539 | .481 | .793 | .787 | .596 |
| Hybrid | .417 | .834 | .790 | .546 | .474 | .821 | .843 | .607 |

We have presented two approaches, based on a very general IOHMM/RNN framework, that achieve state-of-the-art performance in the prediction of proteins contact maps at fine and coarse-grained levels of resolution. In principle both methods can be applied to both resolution levels, although the indirect prediction is computationally too demanding for fine-grained prediction of large proteins. Several extensions are currently under development, including the integration of these methods into complete 3D structure predictors. While these systems require long training periods, once trained they can rapidly sift through large proteomic data sets.

**Acknowledgments**

The work of PB and GP is supported by a Laurel Wilkening Faculty Innovation award and awards from NIH, BREP, Sun Microsystems, and the California Institute for Telecommunications and Information Technology. The work of PF and AV is partially supported by a MURST grant.

# References

[1] D. Baker and A. Sali. Protein structure prediction and structural genomics. *Science*, 294:93–96, 2001.

[2] P. Baldi and S. Brunak and P. Frasconi and G. Soda and G. Pollastri. Exploiting the past and the future in protein secondary structure prediction. *Bioinformatics*, 15(11):937–946, 1999.

[3] P. Baldi and Y. Chauvin. Hybrid modeling, HMM/NN architectures, and protein applications. *Neural Computation*, 8(7):1541–1565, 1996.

[4] P. Baldi and G. Pollastri. Machine learning structural and functional proteomics. *IEEE Intelligent Systems. Special Issue on Intelligent Systems in Biology*, 17(2), 2002.

[5] Y. Bengio and P. Frasconi. Input-output HMM's for sequence processing. *IEEE Trans. on Neural Networks*, 7:1231–1249, 1996.

[6] P. Fariselli, O. Olmea, A. Valencia, and R. Casadio. Prediction of contact maps with neural networks and correlated mutations. *Protein Engineering*, 14:835–843, 2001.

[7] P. Frasconi, M. Gori, and A. Sperduti. A general framework for adaptive processing of data structures. *IEEE Trans. on Neural Networks*, 9:768–786, 1998.

[8] Z. Ghahramani and M. I. Jordan. Factorial hidden Markov models *Machine Learning*, 29:245–273, 1997.

[9] W. Kabsch and C. Sander. Dictionary of protein secondary structure: pattern recognition of hydrogen-bonded and geometrical features. *Biopolymers*, 22:2577–2637, 1983.

[10] A. M. Lesk, L. Lo Conte, and T. J. P. Hubbard. Assessment of novel fold targets in CASP4: predictions of three-dimensional structures, secondary structures, and interresidue contacts. *Proteins*, 45, S5:98–118, 2001.

[11] G. Pollastri and P. Baldi. Predition of contact maps by GIOHMMs and recurrent neural networks using lateral propagation from all four cardinal corners. Proceedings of 2002 ISMB (Intelligent Systems for Molecular Biology) Conference. *Bioinformatics*, 18, S1:62–70, 2002.

[12] G. Pollastri, D. Przybylski, B. Rost, and P. Baldi. Improving the prediction of protein secondary structure in three and eight classes using recurrent neural networks and profiles. *Proteins*, 47:228–235, 2002.

[13] G. Pollastri, P. Baldi, P. Fariselli, and R. Casadio. Prediction of coordination number and relative solvent accessibility in proteins. *Proteins*, 47:142–153, 2002.

[14] M. Vendruscolo, E. Kussell, and E. Domany. Recovery of protein structure from contact maps. *Folding and Design*, 2:295–306, 1997.
